# Value Function Approximation with Diffusion Wavelets and Laplacian Eigenfunctions

**Sridhar Mahadevan**
Department of Computer Science
University of Massachusetts
Amherst, MA 01003
mahadeva@cs.umass.edu

**Mauro Maggioni**
Program in Applied Mathematics
Department of Mathematics
Yale University
New Haven, CT 06511
mauro.maggioni@yale.edu

## Abstract

We investigate the problem of automatically constructing efficient representations or basis functions for approximating value functions based on analyzing the structure and topology of the state space. In particular, two novel approaches to value function approximation are explored based on automatically constructing basis functions on state spaces that can be represented as graphs or manifolds: one approach uses the eigenfunctions of the Laplacian, in effect performing a global Fourier analysis on the graph; the second approach is based on diffusion wavelets, which generalize classical wavelets to graphs using multiscale dilations induced by powers of a diffusion operator or random walk on the graph. Together, these approaches form the foundation of a new generation of methods for solving large Markov decision processes, in which the underlying representation and policies are simultaneously learned.

## 1  Introduction

Value function approximation (VFA) is a well-studied problem: a variety of linear and nonlinear architectures have been studied, which are not automatically derived from the geometry of the underlying state space, but rather handcoded in an *ad hoc* trial-and-error process by a human designer [1]. A new framework for VFA called *proto-reinforcement learning* (PRL) was recently proposed in [7, 8, 9]. Instead of learning task-specific value functions using a handcoded parametric architecture, agents learn proto-value functions, or global basis functions that reflect intrinsic large-scale geometric constraints that all value functions on a manifold [11] or graph [3] adhere to, using spectral analysis of the self-adjoint Laplace operator. This approach also yields new control learning algorithms called *representation policy iteration* (RPI) where both the underlying representations (basis functions) and policies are simultaneously learned. Laplacian eigenfunctions also provide ways of automatically decomposing state spaces since they reflect *bottlenecks* and other global geometric invariants.

In this paper, we extend the earlier Laplacian approach in a new direction using the recently proposed *diffusion wavelet transform* (DWT), which is a compact multi-level representation of Markov diffusion processes on manifolds and graphs [4, 2]. Diffusion wavelets

provide an interesting alternative to global Fourier eigenfunctions for value function approximation, since they encapsulate all the traditional advantages of wavelets: basis functions have compact support, and the representation is inherently hierarchical since it is based on multi-resolution modeling of processes at different spatial and temporal scales.

## 2 Technical Background

This paper uses the framework of spectral graph theory [3] to build basis representations for smooth (value) functions on graphs induced by Markov decision processes. Given any graph $G$, an obvious but poor choice of representation is the "table-lookup" orthonormal encoding, where $\phi(i) = [0 \ldots i \ldots 0]$ is the encoding of the $i^{th}$ node in the graph. This representation does not reflect the topology of the specific graph under consideration. Polynomials are another popular choice of orthonormal basis functions [5], where $\phi(s) = [1 \ s \ldots s^k]$ for some fixed $k$. This encoding has two disadvantages: it is numerically unstable for large graphs, and is dependent on the ordering of vertices. In this paper, we outline a new approach to the problem of building basis functions on graphs using Laplacian eigenfunctions and diffusion wavelets.

A finite Markov decision process (MDP) $M = (S, A, P^a_{ss'}, R^a_{ss'})$ is defined as a finite set of states $S$, a finite set of actions $A$, a transition model $P^a_{ss'}$ specifying the distribution over future states $s'$ when an action $a$ is performed in state $s$, and a corresponding reward model $R^a_{ss'}$ specifying a scalar cost or reward [10]. A state value function is a mapping $S \rightarrow \mathcal{R}$ or equivalently a vector in $\mathcal{R}^{|S|}$. Given a policy $\pi : S \rightarrow A$ mapping states to actions, its corresponding value function $V^\pi$ specifies the expected long-term discounted sum of rewards received by the agent in any given state $s$ when actions are chosen using the policy. Any optimal policy $\pi^*$ defines the same unique optimal value function $V^*$ which satisfies the nonlinear constraints

$$V^*(s) = \max_a \sum_{s'} P^a_{ss'} \left( R^a_{ss'} + \gamma V^*(s') \right)$$

For any MDP, any policy induces a Markov chain that partitions the states into classes: transient states are visited initially but not after a finite time, and recurrent states are visited infinitely often. In *ergodic* MDPs, the set of transient states is empty. The construction of basis functions below assumes that the Markov chain induced by a policy is a reversible random walk on the state space. While some policies may not induce such Markov chains, the set of basis functions learned from a reversible random walk can still be useful in approximating value functions for (reversible or non-reversible) policies. In other words, the construction of the basis functions can be considered an *off-policy* method: just as in Q-learning where the exploration policy differs from the optimal learned policy, in the proposed approach the actual MDP dynamics may induce a different Markov chain than the one analyzed to build representations. Reversible random walks greatly simplify spectral analysis since such random walks are similar to a symmetric operator on the state space.

### 2.1 Smooth Functions on Graphs and Value Function Representation

We assume the state space can be modeled as a finite undirected weighted graph $(G, E, W)$, but the approach generalizes to Riemannian manifolds. We define $x \sim y$ to mean an edge between $x$ and $y$, and the degree of $x$ to be $d(x) = \sum_{x \sim y} w(x, y)$. $D$ will denote the diagonal matrix defined by $D_{xx} = d(x)$, and $W$ the matrix defined by $W_{xy} = w(x, y) = w(y, x)$. The $\mathcal{L}^2$ norm of a function on $G$ is $||f||_2^2 = \sum_{x \in G} |f(x)|^2 d(x)$. The gradient of a function is $\nabla f(i, j) = w(i, j)(f(i) - f(j))$ if there is an edge $e$ connecting $i$ to $j$, 0 otherwise. The smoothness of a function on a graph, can be measured by the Sobolev norm

$$||f||^2_{\mathcal{H}^2} = ||f||_2^2 + ||\nabla f||_2^2 = \sum_x |f(x)|^2 d(x) + \sum_{x \sim y} |f(x) - f(y)|^2 w(x, y). \quad (1)$$

The first term in this norm controls the size (in terms of $\mathcal{L}^2$-norm) for the function $f$, and the second term controls the size of the gradient. The smaller $||f||_{\mathcal{H}^2}$, the smoother is $f$. We will assume that the value functions we consider have small $\mathcal{H}^2$ norms, except at a few points, where the gradient may be large. Important variations exist, corresponding to different measures on the vertices and edges of $G$.

Classical techniques, such as *value iteration* and *policy iteration* [10], represent value functions using an orthonormal basis $(e_1, \ldots, e_{|S|})$ for the space $\mathcal{R}^{|S|}$ [1]. For a fixed precision $\epsilon$, a value function $V^\pi$ can be approximated as

$$||V^\pi - \sum_{i \in S(\epsilon)} \alpha_i^\pi e_i|| \leq \epsilon$$

with $\alpha_i = < V^\pi, e_i >$ since the $e_i$'s are orthonormal, and the approximation is measured in some norm, such as $\mathcal{L}^2$ or $\mathcal{H}^2$. The goal is to obtain representations in which the index set $S(\epsilon)$ in the summation is as small as possible, for a given approximation error $\epsilon$. This hope is well founded at least when $V^\pi$ is smooth or piecewise smooth, since in this case it should be compressible in some well chosen basis $\{e_i\}$.

# 3   Function Approximation using Laplacian Eigenfunctions

The combinatorial Laplacian $L$ [3] is defined as

$$Lf(x) = \sum_{y \sim x} w(x, y)(f(x) - f(y)) = (D - W)f .$$

Often one considers the *normalized* Laplacian $\mathcal{L} = D^{-\frac{1}{2}}(D-W)D^{-\frac{1}{2}}$ which has spectrum in $[0, 2]$. This Laplacian is related to the notion of smoothness as above, since $\langle f, \mathcal{L}f \rangle = \sum_x f(x) Lf(x) = \sum_{x,y} w(x, y)(f(x) - f(y))^2 = ||\nabla f||_2^2$, which should be compared with (1). Functions that satisfy the equation $\mathcal{L}f = 0$ are called *harmonic*. The Spectral Theorem can be applied to $\mathcal{L}$ (or $L$), yielding a discrete set of eigenvalues $0 \leq \lambda_0 \leq \lambda_1 \leq \ldots \lambda_i \leq \ldots$ and a corresponding orthonormal basis of eigenfunctions $\{\xi_i\}_{i \geq 0}$, solutions to the eigenvalue problem $\mathcal{L}\xi_i = \lambda_i \xi_i$.

The eigenfunctions of the Laplacian can be viewed as an orthonormal basis of global Fourier smooth functions that can be used for approximating any value function on a graph. These basis functions capture large-scale features of the state space, and are particularly sensitive to "bottlenecks", a phenomenon widely studied in Riemannian geometry and spectral graph theory [3]. Observe that $\xi_i$ satisfies $||\nabla \xi_i||_2^2 = \lambda_i$. In fact, the variational characterization of eigenvectors shows that $\xi_i$ is the normalized function orthogonal to $\xi_0, \ldots, \xi_{i-1}$ with minimal $||\nabla \xi_i||_2$. Hence the projection of a function $f$ on $S$ onto the top $k$ eigenvectors of the Laplacian is the smoothest approximation to $f$, in the sense of the norm in $\mathcal{H}^2$. A potential drawback of Laplacian approximation is that it detects only global smoothness, and may poorly approximate a function which is not globally smooth but only piecewise smooth, or with different smoothness in different regions. These drawbacks are addressed in the context of analysis with diffusion wavelets, and in fact partly motivated their construction.

# 4   Function Approximation using Diffusion Wavelets

Diffusion wavelets were introduced in [4, 2], in order to perform a fast multiscale analysis of functions on a manifold or graph, generalizing wavelet analysis and associated signal processing techniques (such as compression or denoising) to functions on manifolds and graphs. They allow the fast and accurate computation of high powers of a Markov chain

```
DiffusionWaveletTree (H_0, Φ_0, J, ε):

// H_0: symmetric conjugate to random walk matrix, represented on the basis Φ_0
// Φ_0 : initial basis (usually Dirac's δ-function basis), one function per column
// J : number of levels to compute
// ε: precision

for j from 0 to J do,

        1. Compute sparse factorization H_j ~_ε Q_j R_j, with Q_j orthogonal.

        2. Φ_{j+1} ← Q_j = H_j R_j^{-1} and [H_0^{2^j}]_{Φ_{j+1}}^{Φ_{j+1}} ~_{jε} H_{j+1} ← R_j R_j^*.

        3. Compute sparse factorization I − Φ_{j+1}Φ_{j+1}^* = Q_j' R_j', with Q_j' orthogonal.

        4. Ψ_{j+1} ← Q_j'.

end
```

Figure 1: Pseudo-code for constructing a Diffusion Wavelet Tree

$P$ on the manifold or graph, including direct computation of the Green's function (or fundamental matrix) of the Markov chain, $(I - P)^{-1}$, which can be used to solve Bellman's equation. Here, "fast" means that the number of operations required is $\mathcal{O}(|S|)$, up to logarithmic factors.

Space constraints permit only a brief description of the construction of diffusion wavelet trees. More details are provided in [4, 2]. The input to the algorithm is a "precision" parameter $\epsilon > 0$, and a weighted graph $(G, E, W)$. We can assume that $G$ is connected, otherwise we can consider each connected component separately. The construction is based on using the natural random walk $P = D^{-1}W$ on a graph and its powers to "dilate", or "diffuse" functions on the graph, and then defining an associated coarse-graining of the graph. We symmetrize $P$ by conjugation and take powers to obtain

$$H^t = D^{\frac{1}{2}} P^t D^{-\frac{1}{2}} = (D^{-\frac{1}{2}} W D^{-\frac{1}{2}})^t = (I - \mathcal{L})^t = \sum_{i \geq 0} (1 - \lambda_i)^t \xi_i(\cdot)\xi_i(\cdot) \qquad (2)$$

where $\{\lambda_i\}$ and $\{\xi_i\}$ are the eigenvalues and eigenfunctions of the Laplacian as above. Hence the eigenfunctions of $H^t$ are again $\xi_i$ and the $i^{\text{th}}$ eigenvalue is $(1 - \lambda_i)^t$. We assume that $H^1$ is a sparse matrix, and that the spectrum of $H^1$ has rapid decay.

A diffusion wavelet tree consist of orthogonal diffusion scaling functions $\Phi_j$ that are smooth bump functions, with some oscillations, at scale roughly $2^j$ (measured with respect to geodesic distance, for small $j$), and orthogonal wavelets $\Psi_j$ that are smooth localized oscillatory functions at the same scale. The scaling functions $\Phi_j$ span a subspace $V_j$, with the property that $V_{j+1} \subseteq V_j$, and the span of $\Psi_j$, $W_j$, is the orthogonal complement of $V_j$ into $V_{j+1}$. This is achieved by using the dyadic powers $H^{2^j}$ as "dilations", to create smoother and wider (always in a geodesic sense) "bump" functions (which represent densities for the symmetrized random walk after $2^j$ steps), and orthogonalizing and downsampling appropriately to transform sets of "bumps" into orthonormal scaling functions.

Computationally (Figure 1), we start with the basis $\Phi_0 = I$ and the matrix $H_0 := H^1$, sparse by assumption, and construct an orthonormal basis of well-localized functions for its range (the space spanned by the columns), up to precision $\epsilon$, through a variation of the Gram-Schmidt orthonormalization scheme, described in [4]. In matrix form, this is a sparse factorization $H_0 \sim_\epsilon Q_0 R_0$, with $Q_0$ orthonormal. Notice that $H_0$ is $|G| \times |G|$, but in general $Q_0$ is $|G| \times |G^{(1)}|$ and $R_0$ is $|G^{(1)}| \times |G|$, with $|G^{(1)}| \leq |G|$. In fact $|G^{(1)}|$ is approximately equal to the number of singular values of $H_0$ larger than $\epsilon$. The

columns of $Q_0$ are an orthonormal basis of scaling functions $\Phi_1$ for the range of $H_0$, written as a linear combination of the initial basis $\Phi_0$. We can now write $H_0^2$ on the basis $\Phi_1$: $H_1 := [H^2]_{\Phi_1}^{\Phi_1} = Q_0^* H_0 H_0 Q_0 = R_0 R_0^*$, where we used $H_0 = H_0^*$. This is a compressed representation of $H_0^2$ acting on the range of $H_0$, and it is a $|G^{(1)}| \times |G^{(1)}|$ matrix. We proceed by induction: at scale $j$ we have an orthonormal basis $\Phi_j$ for the rank of $H^{2^j-1}$ up to precision $j\epsilon$, represented as a linear combination of elements in $\Phi_{j-1}$. This basis contains $|G^{(j)}|$ functions, where $|G^{(j)}|$ is comparable with the number of eigenvalues $\lambda_j$ of $H_0$ such that $\lambda_j^{2^j-1} \geq \epsilon$. We have the operator $H_0^{2^j}$ represented on $\Phi_j$ by a $|G^{(j)}| \times |G^{(j)}|$ matrix $H_j$, up to precision $j\epsilon$. We compute a sparse decomposition of $H_j \sim_\epsilon Q_j R_j$, and obtain the next basis $\Phi_{j+1} = Q_j = H_j R_j^{-1}$ and represent $H_0^{2^{j+1}}$ on this basis by the matrix $H_{j+1} := [H^{2^j}]_{\Phi_{j+1}}^{\Phi_{j+1}} = Q_j^* H_j H_j Q_j = R_j R_j^*$.

Wavelet bases for the spaces $W_j$ can be built analogously by factorizing $I_{V_j} - Q_{j+1} Q_{j+1}^*$, which is the orthogonal projection on the complement of $V_{j+1}$ into $V_j$. The spaces can be further split to obtain wavelet packets [2]. A Fast Diffusion Wavelet Transform allows expanding in $\mathcal{O}(n)$ (where $n$ is the number of vertices) computations any function in the wavelet, or wavelet packet, basis, and efficiently search for the most suitable basis set. Diffusion wavelets and wavelet packets are a very efficient tool for representation and approximation of functions on manifolds and graphs [4, 2], generalizing to these general spaces the nice properties of wavelets that have been so successfully applied to similar tasks in Euclidean spaces.

Diffusion wavelets allow computing $H^{2^k} f$ for any fixed $f$, in order $\mathcal{O}(kn)$. This is non-trivial because while the matrix $H$ is sparse, large powers of it are not, and the computation $H \cdot H \ldots \cdot (H(Hf)) \ldots )$ involves $2^k$ matrix-vector products. As a notable consequence, this yields a fast algorithm for computing the Green's function, or fundamental matrix, associated with the Markov process $H$, via $(I-H^1)^{-1} f = \sum_{k \geq 0} H^k = \prod_{k \geq 0} (I + H^{2^k}) f$. In a similar way one can compute $(I - P)^{-1}$. For large classes of Markov chains we can perform this computation in time $\mathcal{O}(n)$, in a direct (as opposed to iterative) fashion. This is remarkable since in general the matrix $(I - H^1)^{-1}$ is full and only writing down the entries would take time $\mathcal{O}(n^2)$. It is the multiscale compression scheme that allows to efficiently represent $(I - H^1)^{-1}$ in compress form, taking advantage of the smoothness of the entries of the matrix. This is discussed in general in [4]. We use this approach to develop a faster policy evaluation step for solving MDPs described in [6]

## 5  Experiments

Figure 2 contrasts Laplacian eigenfunctions and diffusion wavelet basis functions in a three room grid world environment. Laplacian eigenfunctions were produced by solving $Lf = \lambda f$, where $L$ is the combinatorial Laplacian, whereas diffusion wavelet basis functions were produced using the algorithm described in Figure 1. The input to both methods is an undirected graph, where edges connect states reachable through a single (reversible) action. Such graphs can be easily learned from a sample of transitions, such as that generated by RL agents while exploring the environment in early phases of policy learning. Note how the intrinsic multi-room environment is reflected in the Laplacian eigenfunctions. The Laplacian eigenfunctions are globally defined over the entire state space, whereas diffusion wavelet basis functions are progressively more compact at higher levels, beginning at the lowest level with the table-lookup representation, and converging at the highest level to basis functions similar to Laplacian eigenfunctions. Figure 3 compares the approximations produced in a two-room grid world MDP with 630 states. These experiments illustrate the superiority of diffusion wavelets: in the first experiment (top row), diffusion wavelets handily outperform Laplacian eigenfunctions because the function is highly nonlinear near

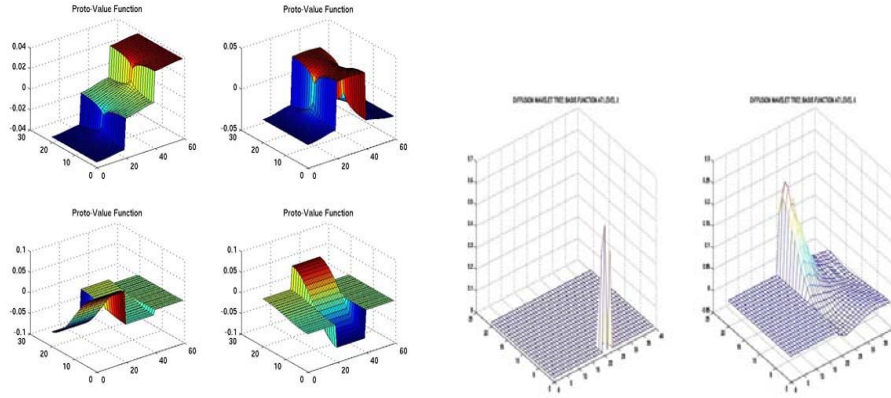

Figure 2: Examples of Laplacian eigenfunctions (left) and diffusion wavelet basis functions (right) computed using the graph Laplacian on a complete undirected graph of a deterministic grid world environment with reversible actions.

the goal, but mostly linear elsewhere. The eigenfunctions contain a lot of ripples in the flat region causing a large residual error. In the second experiment (bottom row), Laplacian eigenfunctions work significantly better because the value function is globally smooth. Even here, the superiority of diffusion wavelets is clear.

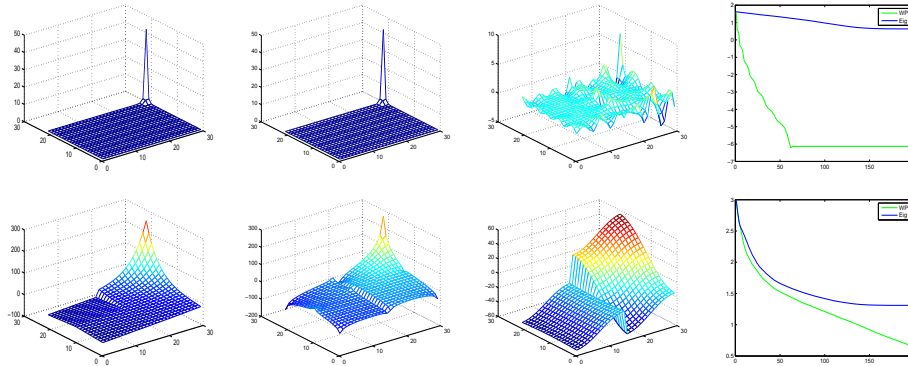

Figure 3: Left column: value functions in a two room grid world MDP, where each room has $21 \times 15$ states connected by a door in the middle of the common wall. Middle two columns: approximations produced by 5 diffusion wavelet bases and Laplacian eigenfunctions. Right column: least-squares approximation error (log scale) using up to 200 basis functions (bottom curve: diffusion wavelets; top curve: Laplacian eigenfunctions). In the top row, the value function corresponds to a random walk. In the bottom row, the value function corresponds to the optimal policy.

## 5.1 Control Learning using Representation Policy Iteration

This section describes results of using the automatically generated basis functions inside a control learning algorithm, in particular the Representation Policy Iteration (RPI) algorithm [8]. RPI is an approximate policy iteration algorithm where the basis functions

$\phi(s, a)$ handcoded in other methods, such as LSPI [5] are learned from a random walk of transitions by computing the graph Laplacian and then computing the eigenfunctions or the diffusion wavelet bases as described above. One striking property of the eigenfunction and diffusion wavelet basis functions is their ability to reflect nonlinearities arising from "bottlenecks" in the state space. Figure 4 contrasts the value function approximation produced by RPI using Laplacian eigenfunctions with that produced by a polynomial approximator. The polynomial approximator yields a value function that is "blind" to the nonlinearities produced by the walls in the two room grid world MDP.

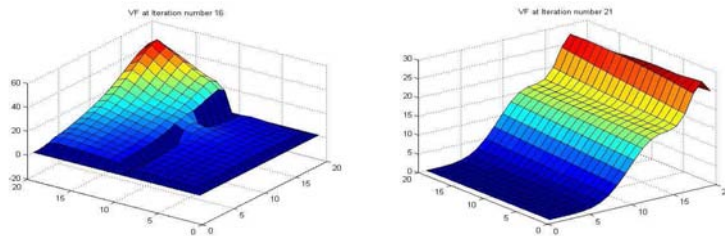

Figure 4: This figures compares the value functions produced by RPI using Laplacian eigenfunctions with that produced by LSPI using a polynomial approximator in a two room grid world MDP with a "bottleneck" region representing the door connecting the two rooms. The Laplacian basis functions on the left clearly capture the nonlinearity arising from the bottleneck, whereas the polynomial approximator on the right smooths the value function across the walls as it is "blind" to the large-scale geometry of the environment.

Table 1 compares the performance of diffusion wavelets and Laplacian eigenfunctions using RPI on the classic chain MDP from [5]. Here, an initial random walk of 5000 steps was carried out to generate the basis functions in a 50 state chain. The chain MDP is a sequential open (or closed) chain of varying number of states, where there are two actions for moving left or right along the chain. In the experiments shown, a reward of 1 was provided in states 10 and 41. Given a fixed $k$, the encoding $\phi(s)$ of a state $s$ for Laplacian eigenfunctions is the vector comprised of the values of the $k^{th}$ lowest-order eigenfunctions on state $k$. For diffusion wavelets, all the basis functions at level $k$ were evaluated at state $s$ to produce the encoding.

| Method | #Trials | Error | Method | #Trials | Error |
|---|---|---|---|---|---|
| RPI DF (5) | 4.4 | 2.4 | LSPI RBF (6) | 3.8 | 20.8 |
| RPI DF (14) | 6.8 | 4.8 | LSPI RBF (14) | 4.4 | 2.8 |
| RPI DF (19) | 8.2 | 0.6 | LSPI RBF (26) | 6.4 | 2.8 |
| RPI Lap (5) | 4.2 | 3.8 | LSPI Poly (5) | 4.2 | 4 |
| RPI Lap (15) | 7.2 | 3 | LSPI Poly (15) | 1 | 34.4 |
| RPI Lap (25) | 9.4 | 2 | LSPI Poly (25) | 1 | 36 |

Table 1: This table compares the performance of RPI using diffusion wavelets and Laplacian eigenfunctions with LSPI using handcoded polynomial and radial basis functions on a 50 state chain graph MDP.

Each row reflects the performance of either RPI using learned basis functions or LSPI with a handcoded basis function (values in parentheses indicate the number of basis functions used for each architecture). The two numbers reported are steps to convergence and the error in the learned policy (number of incorrect actions), averaged over 5 runs. Laplacian and diffusion wavelet basis functions provide a more stable performance at both the low end and at the higher end, as compared to the handcoded basis functions. As the number of

basis functions are increased, RPI with Laplacian basis functions takes longer to converge, but learns a more accurate policy. Diffusion wavelets converge slower as the number of basis functions is increased, giving the best results overall with 19 basis functions. Unlike Laplacian eigenfunctions, the policy error is not monotonically decreasing as the number of bases functions is increased. This result is being investigated. LSPI with RBF is unstable at the low end, converging to a very poor policy for 6 basis functions. LSPI with a 5 degree polynomial approximator works reasonably well, but its performance noticeably degrades at higher degrees, converging to a very poor policy in one step for $k = 15$ and $k = 25$.

## 6 Future Work

We are exploring many extensions of this framework, including extensions to factored MDPs, approximating action value functions as well as large state spaces by exploiting symmetries defined by a group of automorphisms of the graph. These enhancements will facilitate efficient construction of eigenfunctions and diffusion wavelets. For large state spaces, one can randomly subsample the graph, construct the eigenfunctions of the Laplacian or the diffusion wavelets on the subgraph, and then interpolate these functions using the Nyström approximation and related low-rank linear algebraic methods. In experiments on the classic inverted pendulum control task, the Nyström approximation yielded excellent results compared to radial basis functions, learning a more stable policy with a smaller number of samples.

## Acknowledgements

This research was supported in part by a grant from the National Science Foundation IIS-0534999.

## References

[1] D. P. Bertsekas and J. N. Tsitsiklis. *Neuro-Dynamic Programming*. Athena Scientific, Belmont, Massachusetts, 1996.

[2] J. Bremer, R. Coifman, M. Maggioni, and A. Szlam. Diffusion wavelet packets. Technical Report Tech. Rep. YALE/DCS/TR-1304, Yale University, 2004. to appear in Appl. Comp. Harm. Anal.

[3] F. Chung. *Spectral Graph Theory*. American Mathematical Society, 1997.

[4] R. Coifman and M Maggioni. Diffusion wavelets. Technical Report Tech. Rep. YALE/DCS/TR-1303, Yale University, 2004. to appear in Appl. Comp. Harm. Anal.

[5] M. Lagoudakis and R. Parr. Least-squares policy iteration. *Journal of Machine Learning Research*, 4:1107–1149, 2003.

[6] M. Maggioni and S. Mahadevan. Fast direct policy evaluation using multiscale Markov Diffusion Processes. Technical Report Tech. Rep.TR-2005-39, University of Massachusetts, 2005.

[7] S. Mahadevan. Proto-value functions: Developmental reinforcement learning. In *Proceedings of the $22^{nd}$ International Conference on Machine Learning*, 2005.

[8] S. Mahadevan. Representation policy iteration. In *Proceedings of the $21^{st}$ International Conference on Uncertainty in Artificial Intelligence*, 2005.

[9] S. Mahadevan. Samuel meets Amarel: Automating value function approximation using global state space analysis. In *National Conference on Artificial Intelligence (AAAI)*, 2005.

[10] M. L. Puterman. *Markov decision processes*. Wiley Interscience, New York, USA, 1994.

[11] S Rosenberg. *The Laplacian on a Riemannian Manifold*. Cambridge University Press, 1997.
